# Generalization Error and Algorithmic Convergence of Median Boosting

**Balázs Kégl**
Department of Computer Science and Operations Research, University of Montreal
CP 6128 succ. Centre-Ville, Montréal, Canada H3C 3J7
kegl@iro.umontreal.ca

## Abstract

We have recently proposed an extension of ADABOOST to regression that uses the median of the base regressors as the final regressor. In this paper we extend theoretical results obtained for ADABOOST to median boosting and to its localized variant. First, we extend recent results on efficient margin maximizing to show that the algorithm can converge to the maximum achievable margin within a preset precision in a finite number of steps. Then we provide confidence-interval-type bounds on the generalization error.

## 1 Introduction

In a recent paper [1] we introduced MEDBOOST, a boosting algorithm that trains base regressors and returns their weighted *median* as the final regressor. In another line of research, [2, 3] extended ADABOOST to boost *localized* or *confidence-rated* experts with input-dependent weighting of the base classifiers. In [4] we propose a synthesis of the two methods, which we call LOCMEDBOOST. In this paper we analyze the algorithmic convergence of MEDBOOST and LOCMEDBOOST, and provide bounds on the generalization error.

We start by describing the algorithm in its most general form, and extend the result of [1] on the convergence of the *robust (marginal)* training error (Section 2). The robustness of the regressor is measured in terms of the dispersion of the expert population, and with respect to the underlying average confidence estimate. In Section 3, we analyze the algorithmic convergence. In particular, we extend recent results [5] on efficient margin maximizing to show that the algorithm can converge to the maximum achievable margin within a preset precision in a finite number of steps. In Section 4, we provide confidence-interval-type bounds on the generalization error by generalizing results obtained for ADABOOST [6, 2, 3]. As in the case of ADABOOST, the bounds justify the algorithmic objective of minimizing the robust training error. Note that the omitted proofs can be found in [4].

## 2 The LOCMEDBOOST algorithm and the convergence result

For the formal description, let the training data be $D_n = ((\mathbf{x}_1, y_1), \ldots, (\mathbf{x}_n, y_n))$ where data points $(\mathbf{x}_i, y_i)$ are from the set $\mathbb{R}^d \times \mathbb{R}$. The algorithm maintains a weight distribution $\mathbf{w}^{(t)} = (w_1^{(t)}, \ldots, w_n^{(t)})$ over the data points. The weights are initialized uniformly

$$\text{LocMedBoost}(D_n, C_\epsilon(y', y), \text{Base}(D_n, \mathbf{w}), \varrho, T)$$

1      $\mathbf{w} \leftarrow (1/n, \ldots, 1/n)$

2      **for** $t \leftarrow 1$ **to** $T$

3          $(h^{(t)}, \kappa^{(t)}) \leftarrow \text{Base}(D_n, \mathbf{w})$      $\triangleright$ *see (1)*

4          **for** $i \leftarrow 1$ **to** $n$

5              $\theta_i \leftarrow 1 - 2C_\epsilon\big(h^{(t)}(\mathbf{x}_i), y_i\big)$      $\triangleright$ *base rewards*

6              $\kappa_i \leftarrow \kappa^{(t)}(\mathbf{x}_i)$      $\triangleright$ *base confidences*

7          $\alpha^{(t)} \leftarrow \underset{\alpha}{\arg\min} \; e^{\varrho\alpha} \sum_{i=1}^{n} w_i^{(t)} e^{-\alpha\kappa_i\theta_i}$

8          **if** $\alpha^{(t)} = \infty$      $\triangleright \kappa_i\theta_i \geq \varrho$ *for all* $i = 1, \ldots, n$

9              **return** $f^{(t)}(\cdot) = \text{med}_{\boldsymbol{\alpha}, \boldsymbol{\kappa}(\cdot)}\big(\mathbf{h}(\cdot)\big)$

10         **if** $\alpha^{(t)} < 0$      $\triangleright$ *equivalent to* $\sum_{i=1}^{n} w_i^{(t)}\kappa_i\theta_i < \varrho$

11             **return** $f^{(t-1)}(\cdot) = \text{med}_{\boldsymbol{\alpha}, \boldsymbol{\kappa}(\cdot)}\big(\mathbf{h}(\cdot)\big)$

12         **for** $i \leftarrow 1$ **to** $n$

13             $w_i^{(t+1)} \leftarrow w_i^{(t)} \dfrac{\exp(-\alpha^{(t)}\kappa_i\theta_i)}{\sum_{j=1}^{n} w_j^{(t)}\exp(-\alpha^{(t)}\kappa_j\theta_j)} = w_i^{(t)}\dfrac{\exp(-\alpha^{(t)}\kappa_i\theta_i)}{Z^{(t)}}$

14     **return** $f^{(T)}(\cdot) = \text{med}_{\boldsymbol{\alpha}, \boldsymbol{\kappa}(\cdot)}\big(\mathbf{h}(\cdot)\big)$

Figure 1: The pseudocode of the LocMedBoost algorithm. $D_n$ is the training data, $C_\epsilon(y', y) \geq I_{\{|y-y'|>\epsilon\}}$ is the cost function, $\text{Base}(D_n, \mathbf{w})$ is the base regression algorithm, $\varrho$ is the robustness parameter, and $T$ is the number of iterations.

in line 1, and are updated in each iteration in line 13 (Figure 1). We suppose that we are given a *base learner* algorithm $\text{Base}(D_n, \mathbf{w})$ that, in each iteration $t$, returns a *base hypothesis* that consists of a real-valued *base regressor* $h^{(t)} \in \mathcal{H}$ and a non-negative *base confidence function* $\kappa^{(t)} \in \mathcal{K}$. In general, the base learner should attempt to minimize the *base objective*

$$e_1^{(t)}(D_n) = 2\sum_{i=1}^{n} w_i^{(t)}\kappa^{(t)}(\mathbf{x}_i)C_\epsilon\big(h^{(t)}(\mathbf{x}_i), y_i\big) - \bar{\kappa}^{(t)}, \tag{1}$$

where $C_\epsilon(y, y')$ is an $\epsilon$-*dependent loss function* satisfying

$$C_\epsilon(y, y') \geq C_\epsilon^{(0-1)}(y, y') = I\{|y - y'| > \epsilon\}, {}^{[1]} \tag{2}$$

and

$$\bar{\kappa}^{(t)} = \sum_{i=1}^{n} w_i\kappa^{(t)}(\mathbf{x}_i) \tag{3}$$

is the *average confidence* of $\kappa^{(t)}$ on the training set. Intuitively, $e_1^{(t)}(D_n)$ is a mixture of the two objectives of error minimization and confidence maximization. The first term is a weighted regression loss where the weight of a point $\mathbf{x}_i$ is the product of its "constant" weight $w_i^{(t)}$ and the confidence $\kappa^{(t)}(\mathbf{x}_i)$ of the base hypothesis. Minimizing this

term means to place the high-confidence region of the base regressor into areas where the regression error is small. On the other hand, the minimization of the second term drives the high-confidence region of the base regressor into dense areas. After Theorem 1, we will explain the derivation of the base objective (1).

To simplify the notation in Figure 1 and in Theorem 1 below, we define the *base rewards* $\theta_i^{(t)}$ and the *base confidences* $\kappa_i^{(t)}$ for each training point $(\mathbf{x}_i, y_i)$, $i = 1, \ldots, n$, base regressor $h^{(t)}$, and base confidence function $\kappa^{(t)}$, $t = 1, \ldots, T$, as

$$\theta_i^{(t)} = 1 - 2C_\epsilon(h^{(t)}(\mathbf{x}_i), y_i) \quad \text{and} \quad \kappa_i^{(t)} = \kappa^{(t)}(\mathbf{x}_i), \tag{4}$$

respectively.[2]

After computing the base rewards and the base confidences in lines 5 and 6, the algorithm sets the weight $\alpha^{(t)}$ of the base regressor $h^{(t)}$ to the value that minimizes the *exponential loss*

$$E_\varrho^{(t)}(\alpha) = e^{\varrho\alpha} \sum_{i=1}^{n} w_i^{(t)} e^{-\alpha \kappa_i \theta_i}, \tag{5}$$

where $\varrho$ is a *robustness* parameter that has a role in keeping the algorithm in its operating range, in avoiding over- and underfitting, and in maximizing the margin (Section 3). If $\kappa_i \theta_i \geq \varrho$ for all training points, then $\alpha^{(t)} = \infty$ and $E_\varrho^{(t)}(\alpha^{(t)}) = 0$, so the algorithm returns the actual regressor (line 9). Intuitively, this means that the capacity of the set of base hypotheses is too large, so we are overfitting. If $\alpha^{(t)} < 0$, the algorithm returns the regressor up to the last iteration (line 11). Intuitively, this means that the capacity of the set of base hypotheses is too small, so we cannot find a new base regressor that would decrease the training loss. In general, $\alpha^{(t)}$ can be found easily by line-search because of the convexity of $E_\varrho^{(t)}(\alpha)$. In some special cases, $\alpha^{(t)}$ can be computed analytically.

In lines 9, 11, or 14, the algorithm returns the weighted median of the base regressors. For the analysis of the algorithm, we formally define the final regressor in a more general manner. First, let $\widetilde{\alpha}^{(t)} = \frac{\alpha^{(t)}}{\sum_{j=1}^{T} \alpha^{(j)}}$ be the normalized coefficient of the base hypothesis $(h^{(t)}, \kappa^{(t)})$, and let

$$c^{(T)}(\mathbf{x}) = \sum_{t=1}^{T} \widetilde{\alpha}^{(t)} \kappa^{(t)}(\mathbf{x}) = \frac{\sum_{t=1}^{T} \alpha^{(t)} \kappa^{(t)}(\mathbf{x})}{\sum_{t=1}^{T} \alpha^{(t)}} \tag{6}$$

be the *average confidence function*[3] after the $T$th iteration. Let $f_{\rho+}^{(T)}(\mathbf{x})$ and $f_{\rho-}^{(T)}(\mathbf{x})$ be the *weighted* $\left(\frac{1+\rho/c^{(T)}(\mathbf{x})}{2}\right)$- and $\left(\frac{1-\rho/c^{(T)}(\mathbf{x})}{2}\right)$-*quantiles*, respectively, of the base regressors $h^{(1)}(\mathbf{x}), \ldots, h^{(T)}(\mathbf{x})$ with respective weights $\alpha^{(1)} \kappa^{(1)}(\mathbf{x}), \ldots, \alpha^{(T)} \kappa^{(T)}(\mathbf{x})$ (Figure 2(a)). Formally, for any $\rho \in \mathbb{R}$, if $-c^{(T)}(\mathbf{x}) < \rho < c^{(T)}(\mathbf{x})$, let

$$f_{\rho+}^{(T)}(\mathbf{x}) = \min_{j} \left\{ h^{(j)}(\mathbf{x}) : \frac{\sum_{t=1}^{T} \alpha^{(t)} \kappa^{(t)}(\mathbf{x}) I\{h^{(j)}(\mathbf{x}) < h^{(t)}(\mathbf{x})\}}{\sum_{t=1}^{T} \alpha^{(t)} \kappa^{(t)}(\mathbf{x})} < \frac{1 - \frac{\rho}{c^{(T)}(\mathbf{x})}}{2} \right\}, \tag{7}$$

$$f_{\rho-}^{(T)}(\mathbf{x}) = \max_{j} \left\{ h^{(j)}(\mathbf{x}) : \frac{\sum_{t=1}^{T} \alpha^{(t)} \kappa^{(t)}(\mathbf{x}) I\{h^{(j)}(\mathbf{x}) > h^{(t)}(\mathbf{x})\}}{\sum_{t=1}^{T} \alpha^{(t)} \kappa^{(t)}(\mathbf{x})} < \frac{1 - \frac{\rho}{c^{(T)}(\mathbf{x})}}{2} \right\}, \tag{8}$$

otherwise (including the case when $c^{(T)}(\mathbf{x}) = 0$) let $f_{\rho+}^{(T)}(\mathbf{x}) = \rho \cdot (+\infty)$ and $f_{\rho-}^{(T)}(\mathbf{x}) = \rho \cdot (-\infty)$[4]. Then the *weighted median* is defined as $f^{(T)}(\cdot) = \mathrm{med}_{\boldsymbol{\alpha}, \boldsymbol{\kappa}(\cdot)}\big(\mathbf{h}(\cdot)\big) = f_{0+}^{(T)}(\cdot)$.

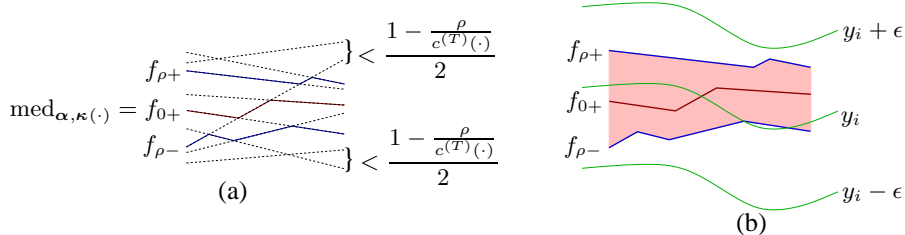

(a)

(b)

Figure 2: (a) Weighted $\left(\frac{1+\rho/c^{(T)}(\mathbf{x})}{2}\right)$- and $\left(\frac{1-\rho/c^{(T)}(\mathbf{x})}{2}\right)$-quantiles, and the weighted median of linear base regressors with equal weights $\alpha^{(t)} = 1/9$, constant base confidence functions $\kappa(\mathbf{x}) \equiv 1$, and $\frac{\rho}{c^{(T)}(\mathbf{x})} \equiv 0.25$. (b) $\rho$-robust $\epsilon$-precise regressor.

To assess the final regressor $f^{(T)}(\cdot)$, we say that $f^{(T)}(\cdot)$ is $\rho$-*robust* $\epsilon$-*precise* on $(\mathbf{x}_i, y_i)$ if and only if $f_{\rho+}^{(T)}(\mathbf{x}_i) \le y_i + \epsilon$, and $f_{\rho-}^{(T)}(\mathbf{x}_i) \ge y_i - \epsilon$. For $\rho \ge 0$, this condition is equivalent to both quantiles being in the "$\epsilon$-*tube*" around $y_i$ (Figure 2(b)).

In the rest of this section we show that the algorithm minimizes the relative frequency of training points on which $f^{(T)}(\cdot)$ is not $\varrho$-robust $\epsilon$-precise. Formally, let the $\rho$-*robust* $\epsilon$-*precise training error* of $f^{(T)}$ be defined as

$$L^{(\rho)}(f^{(T)}) = \frac{1}{n} \sum_{i=1}^{n} I\left\{ f_{\rho+}^{(T)}(\mathbf{x}_i) > y_i + \epsilon \ \ \vee \ \ f_{\rho-}^{(T)}(\mathbf{x}_i) < y_i - \epsilon \right\}.^5 \qquad (9)$$

If $\rho = 0$, $L^{(0)}(f^{(T)})$ gives the relative frequency of training points on which the regressor $f^{(T)}$ has a larger $L_1$ error than $\epsilon$. If we have equality in (2), this is exactly the average loss of the regressor $f^{(T)}$ on the training data. A small value for $L^{(0)}(f^{(T)})$ indicates that the regressor predicts most of the training points with $\epsilon$-precision, whereas a small value for $L^{(\rho)}(f^{(T)})$ with a positive $\rho$ suggests that the prediction is not only precise but also robust in the sense that a small perturbation of the base regressors and their weights will not increase $L^{(0)}(f^{(T)})$. For classification with bi-valued base classifiers $h : \mathbb{R}^d \mapsto \{-1, 1\}$, the definition (9) (with $\epsilon = 1$) recovers the traditional notion of robust training error, that is, $L^{(\rho)}(f^{(T)})$ is the relative frequency of data points with *margin* smaller than $\rho$.

The following theorem upper bounds the $\rho$-robust $\epsilon$-precise training error $L^{(\rho)}$ of the regressor $f^{(T)}$ output by LocMedBoost.

**Theorem 1** *Let $L^{(\rho)}(f^{(T)})$ defined as in (9) and suppose that condition (2) holds for the loss function $C_\epsilon(\cdot, \cdot)$. Define the base rewards $\theta_i^{(t)}$ and the base confidences $\kappa_i^{(t)}$ as in (4). Let $w_i^{(t)}$ be the weight of training point $\mathbf{x}_i$ after the $t$th iteration (updated in line 13 in Figure 1), and let $\alpha^{(t)}$ be the weight of the base regressor $h^{(t)}(\cdot)$ (computed in line 7 in Figure 1). Then for all $\rho \in \mathbb{R}$*

$$L^{(\rho)}(f^{(T)}) \le \prod_{t=1}^{T} E_\rho^{(t)}(\alpha^{(t)}), \qquad (10)$$

*where $E_\rho^{(t)}(\alpha^{(t)})$ is defined in (5).*

The proof is based on the observation that if the median of the base regressors goes further than $\epsilon$ from the real response $y_i$ at training point $\mathbf{x}_i$, then most of the base regressors must also be far from $y_i$, giving small base rewards to this point.

The goal of LOCMEDBOOST is to minimize $L^{(\rho)}(f^{(T)})$ at $\rho = \varrho$ so, in view of Theorem 1, our goal in each iteration $t$ is to minimize $E_\varrho^{(t)}$ (5). To derive the base objective (1), we follow the two step functional gradient descent procedure [7], that is, first we maximize the negative gradient $-E_\varrho'(\alpha)$ in $\alpha = 0$, then we do a line search to determine $\alpha^{(t)}$. Using this approach, the base objective becomes $e_1(D_n) = -\sum_{i=1}^{n} w_i^{(t)} \kappa_i \theta_i$, which is identical to (1). Note that since $E_\varrho^{(t)}(\alpha)$ is convex and $E_\varrho^{(t)}(0) = 1$, a positive $\alpha^{(t)}$ means that $\min_\alpha E_\varrho^{(t)}(\alpha) = E_\varrho^{(t)}(\alpha^{(t)}) < 1$, so the condition in line 10 in Figure 1 guarantees that the upper bound of (10) decreases in each step.

## 3 Setting $\varrho$ and maximizing the minimum margin

In practice, ADABOOST works well with $\varrho = 0$, so setting $\varrho$ to a positive value is only an alternative regularization option to early stopping. In the case of LOCMEDBOOST, however, one must carefully choose $\varrho$ to keep the algorithm in its operating range and to avoid over- and underfitting. A too small $\varrho$ means that the algorithm can overfit and stop in line 9. In binary classification this is an unrealistic situation: it means that there is a base classifier that correctly classifies all data points. On the other hand, it can happen easily in the abstaining classifier/regressor model, when $\kappa^{(t)}(\mathbf{x}) = 0$ on a possibly large input region. In this case, a base classifier can correctly classify (or a base regressor can give positive base rewards $\theta_i$ to) all data points on which it does not abstain, so if $\varrho = 0$, the algorithm stops in line 9. At the other end of the spectrum, a large $\varrho$ can make the algorithm underfit and stop in line 11, so one needs to set $\varrho$ carefully in order to avoid early stopping in lines 9 or 11.

From the point of view of generalization, $\varrho$ also has an important role as a regularization parameter. A larger $\varrho$ decreases the stepsize $\alpha^{(t)}$ in the functional gradient view. From another aspect, a larger $\varrho$ decreases the effective capacity of the the class of base hypotheses by restricting the set of admissible base hypotheses to those having small errors. In general, $\varrho$ has a potential role in balancing between over- and underfitting so, in practice, we suggest that it be validated together with the number of iterations $T$ and other possible complexity parameters of the base hypotheses.

In the context of ADABOOST, there have been several proposals to set $\varrho$ in an adaptive way to effectively maximize the minimum margin. In the rest of this section, we extend the analysis of *marginal boosting* [5] to this general case. Although the agressive maximization of the minimum margin can lead to overfitting, the analysis can provide valuable insight into the understanding of LOCMEDBOOST and so it can guide the setting of $\varrho$ in practice.

For the sake of simplicity, let us assume that base hypotheses $(h, \kappa)$ come from a finite set[6] $\mathcal{H}_N$ with cardinality $N$, and let $\mathcal{H}^{(t)} = \{(h^{(1)}, \kappa^{(1)}), \ldots, (h^{(t)}, \kappa^{(t)})\}$ be the set of base hypotheses after the $t$th iteration. Let us define the *edge* of the base hypothesis $(h, \kappa) \in \mathcal{H}_N$ as[7]

$$\gamma_{(h,\kappa)}(\mathbf{w}) = \sum_{i=1}^{n} w_i \kappa_i \theta_i = \sum_{i=1}^{n} w_i \kappa(\mathbf{x}_i)\Big(1 - 2C_\epsilon\big(h(\mathbf{x}_i), y_i\big)\Big),$$

and the *maximum edge* in the $t$th iteration as $\gamma^{*(t)} = \max_{(h,\kappa)\in\mathcal{H}_N} \gamma_{(h,\kappa)}(\mathbf{w}^{(t)})$. Note that $\gamma_{(h,\kappa)}(\mathbf{w}) = -e_1(D_n)$, so with this terminology, the objective of the base learner is

to maximize the edge $\gamma^{(t)} = \gamma_{(h^{(t)}, \kappa^{(t)})}(\mathbf{w}^{(t)})$ (if the maximum is achieved, then $\gamma^{(t)} = \gamma^{*(t)}$), and the algorithm stops in line 11 if the edge $\gamma^{(t)}$ is less than $\varrho$. On the other hand, let us define the *margin* on a point $(\mathbf{x}, y)$ as the average reward[8]

$$\rho_{(\mathbf{x},y)}(\boldsymbol{\alpha}) = \sum_{j=1}^{N} \widetilde{\alpha}^{(j)} \kappa^{(j)} \theta^{(j)} = \sum_{j=1}^{N} \widetilde{\alpha}^{(j)} \kappa^{(j)}(\mathbf{x}) \Big( 1 - 2 C_\epsilon \big( h^{(j)}(\mathbf{x}), y \big) \Big).$$

Let us denote the *minimum margin* over the data points in the $t$th iteration by

$$\rho^{*(t)} = \min_{(\mathbf{x},y) \in D_n} \rho_{(\mathbf{x},y)}(\boldsymbol{\alpha}^{(t-1)}), \tag{11}$$

where $\boldsymbol{\alpha}^{(t-1)} = \big( \alpha^{(1)}, \ldots, \alpha^{(t-1)} \big)$ is the vector of base hypothesis coefficients up to the $(t-1)$th iteration.

It is easy to see that in each iteration, the maximum edge over the base hypotheses is at least the minimum margin over the training points:

$$\gamma^{*(t)} = \max_{(h,\kappa) \in \mathcal{H}_N} \gamma_{(h,\kappa)}(\mathbf{w}^{(t)}) \geq \min_{(\mathbf{x},y) \in D_n} \rho_{(\mathbf{x},y)}(\boldsymbol{\alpha}^{(t-1)}) = \rho^{*(t)}.$$

Moreover, as several authors (e.g., [5]) noted in the context of ADABOOST, by the Min-Max-Theorem of von Neumann [8] we have

$$\gamma^* = \min_{\mathbf{w}} \max_{(h,\kappa) \in \mathcal{H}_N} \gamma_{(h,\kappa)}(\mathbf{w}) = \max_{\boldsymbol{\alpha}} \min_{(\mathbf{x},y) \in D_n} \rho_{(\mathbf{x},y)}(\boldsymbol{\alpha}) = \rho^*,$$

so the minimum achievable maximal edge by any weighting over the training points is equal to the maximum achievable minimal margin by any weighting over the base hypotheses. To converge to $\rho^*$ within a factor $\nu$ in finite time, [5] sets

$$\varrho_{RW}^{(t)} = \min_{j=1,\ldots,t} \gamma^{(j)} - \nu,$$

and shows that $\rho^{*(t)}$ exceeds $\rho^* - \nu$ after $\left\lceil \frac{2 \log n}{\nu^2} \right\rceil + 1$ steps.

In the following, we extend these results to the general case of LOCMEDBOOST. First we define the *minimum and maximum achievable base rewards* by

$$\rho_{\min} = \min_{(h,\kappa) \in \mathcal{H}_N} \min_{(\mathbf{x},y) \in D_n} \kappa(\mathbf{x}) \Big( 1 - 2 C_\epsilon \big( h(\mathbf{x}), y \big) \Big), \tag{12}$$

$$\rho_{\max} = \max_{(h,\kappa) \in \mathcal{H}_N} \max_{(\mathbf{x},y) \in D_n} \kappa(\mathbf{x}) \Big( 1 - 2 C_\epsilon \big( h(\mathbf{x}), y \big) \Big), \tag{13}$$

respectively. Let $A = \rho_{\max} - \rho_{\min}$, $\widetilde{\gamma}^{(t)} = \gamma^{(t)} - \rho_{\min}$, and $\widetilde{\varrho}^{(t)} = \varrho^{(t)} - \rho_{\min}$.[9]

**Lemma 1 (Generalization of Lemma 3 in [5])** *Assume that $\rho_{\min} \leq \varrho^{(t)} \leq \gamma^{(t)}$. Then*

$$E_{\varrho^{(t)}}^{(t)}(\alpha^{(t)}) \leq \exp \left[ -\frac{\widetilde{\varrho}^{(t)}}{A} \log \left( \frac{\widetilde{\varrho}^{(t)}}{\widetilde{\gamma}^{(t)}} \right) - \frac{A - \widetilde{\varrho}^{(t)}}{\varrho^{(t)}} \log \left( \frac{A - \widetilde{\varrho}^{(t)}}{A - \widetilde{\gamma}^{(t)}} \right) \right]. \tag{14}$$

Finite convergence of LOCMEDBOOST both with $\varrho^{(t)} = \varrho = \text{const.}$ and with an adaptive $\varrho^{(t)} = \varrho_{RW}^{(t)}$ is based on the following general result.

**Theorem 2** *Assume that $\varrho^{(t)} \leq \gamma^{(t)} - \nu$. Let $\rho = \sum_{t=1}^{T} \widetilde{\alpha}^{(t)} \varrho^{(t)}$. Then $L^{(\rho)}(f^{(T)}) = 0$ (so $\rho^{*(t)} > \rho$) after at most $T = \left\lceil \frac{A^2 \log n}{2\nu^2} \right\rceil + 1$ iterations.*

The first consequence is the convergence of LOCMEDBOOST with a constant $\varrho$.

**Corollary 1 (Generalization of Corollary 4 in [5])** *Assume that the weak learner always achieves an edge $\gamma^{(t)} \geq \rho^*$. If $\rho_{\min} \leq \varrho < \rho^*$, then $\rho^{*(t)} > \varrho$ after at most $T = \left\lceil \frac{A^2 \log n}{2(\rho^* - \varrho)^2} \right\rceil + 1$ steps.*

The second corollary shows that if $\varrho$ is set adaptively to $\varrho_{RW}^{(t)}$ then the minimum margin $\rho^{*(t)}$ will converge to $\rho^*$ within a precision $\nu$ in a finite number of steps.

**Corollary 2 (Generalization of Theorem 6 in [5])** *Assume that the weak learner always achieves an edge $\gamma^{(t)} \geq \rho^*$. If $\rho_{\min} \leq \varrho^{(t)} = \gamma^{(t)} - \nu, \nu > 0$, then $\rho^{*(t)} > \rho^* - \nu$ after at most $T = \left\lceil \frac{A^2 \log n}{2\nu^2} \right\rceil + 1$ iterations.*

## 4 The generalization error

In this section we extend probabilistic bounds on the generalization error obtained for ADABOOST [6], confidence-rated ADABOOST [2], and localized boosting [3]. Here we suppose that the data set $D_n$ is generated independently according to a distribution $\mathcal{D}$ over $\mathbb{R}^d \times \mathbb{R}$. The results provide bounds on the confidence-interval-type error

$$L(f^{(T)}) = \mathbf{P}_{\mathcal{D}} \left[ \left| f^{(T)}(X) - Y \right| > \epsilon \right],$$

where $(X, Y)$ is a random point generated according to $\mathcal{D}$ independently from points in $D_n$. The bounds state that with a large probability,

$$L(f^{(T)}) < L^{(\rho)}(f^{(T)}) + C(n, \rho, \mathcal{H}, \mathcal{K}),$$

where the complexity term $C$ depends on the size or the pseudo-dimension of the base regressor set $\mathcal{H}$, and the smoothness of the base confidence functions in $\mathcal{K}$. As in the case of ADABOOST, these bounds qualitatively justify the minimization of the robust training error $L^{(\rho)}(f^{(T)})$.

Let $\mathcal{C}$ be the set of combined regressors obtained as a weighted median of base regressors from $\mathcal{H}$, that is,

$$\mathcal{C} = \left\{ f(\cdot) = \mathrm{med}_{\boldsymbol{\alpha}, \boldsymbol{\kappa}(\cdot)}(\mathbf{h}(\cdot)) \big| \mathbf{h} \in \mathcal{H}^N, \boldsymbol{\alpha} \in \mathbb{R}^{+N}, \boldsymbol{\kappa} \in \mathcal{K}^N, N \in \mathbb{Z}^+ \right\}.$$

In the simplest case, we assume that $\mathcal{H}$ is finite and base coefficients are constant.

**Theorem 3 (Generalization of Theorem 1 in [6])** *Let $\mathcal{D}$ be a distribution over $\mathbb{R}^d \times \mathbb{R}$, and let $D_n$ be a sample of $n$ points generated independently at random according to $\mathcal{D}$. Assume that the base regressor set $\mathcal{H}$ is finite, and $\mathcal{K}$ contains only $\kappa(\mathbf{x}) \equiv 1$. Then with probability $1 - \delta$ over the random choice of the training set $D_n$, any $f \in \mathcal{C}$ satisfies the following bound for all $\rho > 0$:*

$$L(f) < L^{(\rho)}(f) + O\left( \frac{1}{\sqrt{n}} \left( \frac{\log n \log |\mathcal{H}|}{\rho^2} + \log \frac{1}{\delta} \right)^{1/2} \right).$$

Similarly to the proof of Theorem 1 in [6], we construct a set $\mathcal{C}_N$ that contains *unweighted* medians of $N$ base functions from $\mathcal{H}$, then approximate $f$ by $g(\cdot) = \mathrm{med}_{\mathbf{1}}(h_1(\cdot), \ldots, h_N(\cdot)) \in \mathcal{C}_N$ where the base functions $h_i$ are selected randomly according to the coefficient distribution $\widetilde{\boldsymbol{\alpha}}$. We then separate the one-sided error into two terms by

$$\mathbf{P}_{\mathcal{D}} \big[ f(X) > Y + \epsilon \big] \leq \mathbf{P}_{\mathcal{D}} \big[ g_{\frac{\rho}{2}+}(X) > Y + \epsilon \big] + \mathbf{P}_{\mathcal{D}} \big[ g_{\frac{\rho}{2}+}(X) \leq Y + \epsilon \big| f(X) > Y + \epsilon \big],$$

and then upper bound the two terms as in [6].

The second theorem extends the first to the case of infinite base regressor sets.

**Theorem 4 (Generalization of Theorem 2 of [6])** *Let $\mathcal{D}$ be a distribution over $\mathbb{R}^d \times \mathbb{R}$, and let $D_n$ be a sample of $n$ points generated independently at random according to $\mathcal{D}$. Assume that the base regressor set $\mathcal{H}$ has pseudodimension $p$, and $\mathcal{K}$ contains only $\kappa(\mathbf{x}) \equiv 1$. Then with probability $1 - \delta$ over the random choice of the training set $D_n$, any $f \in \mathcal{C}$ satisfies the following bound for all $\rho > 0$:*

$$L(f) < L^{(\rho)}(f) + O\left(\frac{1}{\sqrt{n}}\left(\frac{p\log^2(n/p)}{\rho^2} + \log\frac{1}{\delta}\right)^{1/2}\right).$$

The proof goes as in Theorem 3 and in Theorem 2 in [6] until we upper bound the shatter coefficient of the set $\mathcal{A} = \left\{\{(\mathbf{x},y): g_{\frac{\rho}{2}+}(\mathbf{x}) > y + \epsilon\}: g \in \mathcal{C}_N, \rho = 0, \frac{4}{N}, \ldots, \frac{2N}{N}\right\}$ by $(N/2+1)(en/p)^{pN}$ where $p$ is the pseudodimension of $\mathcal{H}$ (or the VC dimension of $\mathcal{H}_+ = \left\{\{(\mathbf{x},y): h(\mathbf{x}) > y\}: h \in \mathcal{H}\right\}$).

In the most general case $\mathcal{K}$ can contain smooth functions.

**Theorem 5 (Generalization of Theorem 1 of [3])** *Let $\mathcal{D}$ be a distribution over $\mathbb{R}^d \times \mathbb{R}$, and let $D_n$ be a sample of $n$ points generated independently at random according to $\mathcal{D}$. Assume that the base regressor set $\mathcal{H}$ has pseudodimension $p$, and $\mathcal{K}$ contains functions $\kappa(\mathbf{x})$ which are lower bounded by a constant $a$, and which satisfy for all $\mathbf{x}, \mathbf{x}' \in \mathbb{R}^d$ the Lipschitz condition $|\kappa(\mathbf{x}) - \kappa(\mathbf{x}')| \le L\|\mathbf{x} - \mathbf{x}'\|_\infty$. Then with probability $1 - \delta$ over the random choice of the training set $D_n$, any $f \in \mathcal{C}$ satisfies the following bound for all $\rho > 0$:*

$$L(f) < L^{(\rho)}(f) + O\left(\frac{1}{\sqrt{n}}\left(\frac{(L/(a\rho))^d p\log^2(n/p)}{\rho^2} + \log\frac{1}{\delta}\right)^{1/2}\right).$$

## 5 Conclusion

In this paper we have analyzed the algorithmic convergence of LOCMEDBOOST by generalizing recent results on efficient margin maximization, and provided bounds on the generalization error by extending similar bounds obtained for ADABOOST.

## Footnotes

[1] The indicator function $I\{A\}$ is 1 if its argument $A$ is true and 0 otherwise.

[2]Note that we will omit the iteration index $^{(t)}$ where it does not cause confusion.

[3]Not to be confused with $\bar{\kappa}^{(t)}$ in (3) which is the average *base* confidence *over the training data*.

[4]In the degenerative case we define $0 \cdot \infty = 0/0 = \infty$.

[5]For the sake of simplicity, in the notation we suppress the fact that $L^{(\rho)}$ depends on the whole sequence of base regressors, base confidences, and weights, not only on the final regressor $f^{(T)}$.

[6]The analysis can be extended to infinite base sets along the lines of [5].

[7]For the sake of simplicity, in the notation we suppress the dependence of $\gamma_{(h,\kappa)}$ on $D_n$.

[8]For the sake of simplicity, in the notation we suppress the dependence of $\rho_{(\mathbf{x},y)}$ on $\mathcal{H}_N$.

[9]In binary classification, $\rho_{\min} = -1$, $\rho_{\max} = 1$, $A = 2$, $\widetilde{\gamma}^{(t)} = 1 + \gamma^{(t)}$, and $\widetilde{\varrho}^{(t)} = 1 + \varrho^{(t)}$.

## References

[1] B. Kégl, "Robust regression by boosting the median," in *Proceedings of the 16th Conference on Computational Learning Theory*, Washington, D.C., 2003, pp. 258–272.

[2] R. E. Schapire and Y. Singer, "Improved boosting algorithms using confidence-rated predictions," *Machine Learning*, vol. 37, no. 3, pp. 297–336, 1999.

[3] R. Meir, R. El-Yaniv, and S. Ben-David, "Localized boosting," in *Proceedings of the 13th Annual Conference on Computational Learning Theory*, 2000, pp. 190–199.

[4] B. Kégl, "Confidence-rated regression by boosting the median," Tech. Rep. 1241, Department of Computer Science, University of Montreal, 2004.

[5] G. Rätsch and M. K. Warmuth, "Efficient margin maximizing with boosting," *Journal of Machine Learning Research (submitted)*, 2003.

[6] R. E. Schapire, Y. Freund, P. Bartlett, and W. S. Lee, "Boosting the margin: a new explanation for the effectiveness of voting methods," *Annals of Statistics*, vol. 26, no. 5, pp. 1651–1686, 1998.

[7] L. Mason, P. Bartlett, J. Baxter, and M. Frean, "Boosting algorithms as gradient descent," in *Advances in Neural Information Processing Systems*. 2000, vol. 12, pp. 512–518, The MIT Press.

[8] J. von Neumann, "Zur Theorie der Gesellschaftsspiele," *Math. Ann.*, vol. 100, pp. 295–320, 1928.
